# Generating Accurate and Diverse Members of a Neural-Network Ensemble

**David W. Opitz**
Computer Science Department
University of Minnesota
Duluth, MN 55812
opitz@d.umn.edu

**Jude W. Shavlik**
Computer Sciences Department
University of Wisconsin
Madison, WI 53706
shavlik@cs.wisc.edu

## Abstract

Neural-network ensembles have been shown to be very accurate classification techniques. Previous work has shown that an effective ensemble should consist of networks that are not only highly correct, but ones that make their errors on different parts of the input space as well. Most existing techniques, however, only indirectly address the problem of creating such a set of networks. In this paper we present a technique called ADDEMUP that uses genetic algorithms to directly search for an accurate and diverse set of trained networks. ADDEMUP works by first creating an initial population, then uses genetic operators to continually create new networks, keeping the set of networks that are as accurate as possible while disagreeing with each other as much as possible. Experiments on three DNA problems show that ADDEMUP is able to generate a set of trained networks that is more accurate than several existing approaches. Experiments also show that ADDEMUP is able to effectively incorporate prior knowledge, if available, to improve the quality of its ensemble.

## 1 Introduction

Many researchers have shown that simply combining the output of many classifiers can generate more accurate predictions than that of any of the individual classifiers (Clemen, 1989; Wolpert, 1992). In particular, combining separately trained neural networks (commonly referred to as a neural-network *ensemble*) has been demonstrated to be particularly successful (Alpaydin, 1993; Drucker et al., 1994; Hansen and Salamon, 1990; Hashem et al., 1994; Krogh and Vedelsby, 1995; Maclin and Shavlik, 1995; Perrone, 1992). Both theoretical (Hansen and Salamon, 1990; Krogh and Vedelsby, 1995) and empirical (Hashem et al., 1994;

Maclin and Shavlik, 1995) work has shown that a good ensemble is one where the individual networks are both accurate and make their errors on different parts of the input space; however, most previous work has either focussed on combining the output of multiple trained networks or only indirectly addressed how we should generate a good set of networks. We present an algorithm, ADDEMUP (Accurate anD Diverse Ensemble-Maker giving United Predictions), that uses genetic algorithms to *generate* a population of neural networks that are highly accurate, while at the same time having minimal overlap on where they make their error.

Traditional ensemble techniques generate their networks by randomly trying different topologies, initial weight settings, parameters settings, or use only a part of the training set in the hopes of producing networks that disagree on where they make their errors (we henceforth refer to *diversity* as the measure of this disagreement). We propose instead to actively *search* for a good set of networks. The key idea behind our approach is to consider many networks and keep a subset of the networks that minimizes our objective function consisting of both an accuracy and a diversity term. In many domains we care more about generalization performance than we do about generating a solution quickly. This, coupled with the fact that computing power is rapidly growing, motivates us to effectively utilize available CPU cycles by continually considering networks to possibly place in our ensemble.

ADDEMUP proceeds by first creating an initial set of networks, then continually produces new individuals by using the genetic operators of crossover and mutation. It defines the overall fitness of an individual to be a combination of accuracy and diversity. Thus ADDEMUP keeps as its population a set of highly fit individuals that will be highly accurate, while making their mistakes in a different part of the input space. Also, it actively tries to generate good candidates by emphasizing the current population's erroneous examples during backpropagation training. Experiments reported herein demonstrate that ADDEMUP is able to generate an effective set of networks for an ensemble.

## 2    The Importance of an Accurate and Diverse Ensemble

Figure 1 illustrates the basic framework of a neural-network ensemble. Each network in the ensemble (network 1 through network $N$ in this case) is first trained using the training instances. Then, for each example, the predicted output of each of these networks ($o_i$ in Figure 1) is combined to produce the output of the ensemble ($\hat{o}$ in Figure 1). Many researchers (Alpaydin, 1993; Hashem et al., 1994; Krogh and Vedelsby, 1995; Mani, 1991) have demonstrated the effectiveness of combining schemes that are simply the weighted average of the networks (i.e., $\hat{o} = \sum_{i \in N} w_i \cdot o_i$ and $\sum_{i \in N} w_i = 1$), and this is the type of ensemble we focus on in this paper.

Hansen and Salamon (1990) proved that for a neural-network ensemble, if the average error rate for a pattern is less than 50% and the networks in the ensemble are independent in the production of their errors, the expected error for that pattern can be reduced to zero as the number of networks combined goes to infinity; however, such assumptions rarely hold in practice. Krogh and Vedelsby (1995) later proved that if diversity[1] $D_i$ of network $i$ is measured by:

$$D_i = \sum_x [o_i(x) - \hat{o}(x)]^2,  \tag{1}$$

then the ensemble generalization error ($\hat{E}$) consists of two distinct portions:

$$\hat{E} = \bar{E} - \bar{D},  \tag{2}$$

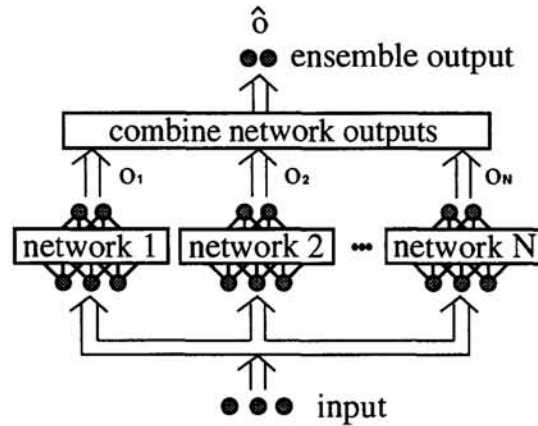

Figure 1: A neural-network ensemble.

where $\bar{D} = \sum_i w_i \cdot D_i$ and $\bar{E} = \sum_i w_i \cdot E_i$ ($E_i$ is the error rate of network $i$ and the $w_i$'s sum to 1). What the equation shows then, is that we want our ensemble to consist of highly correct networks that disagree as much as possible. *Creating such a set of networks is the focus of this paper.*

## 3  The ADDEMUP Algorithm

Table 1 summarizes our new algorithm, ADDEMUP, that uses genetic algorithms to generate a set of neural networks that are accurate and diverse in their classifications. (Although ADDEMUP currently uses neural networks, it could be easily extended to incorporate other types of learning algorithms as well.) ADDEMUP starts by creating and training its initial population of networks. It then creates new networks by using standard genetic operators, such as crossover and mutation. ADDEMUP trains these new individuals, emphasizing examples that are misclassified by the current population, as explained below. ADDEMUP adds these new networks to the population then scores each population members with the fitness function:

$$Fitness_i = Accuracy_i + \lambda \ Diversity_i = (1 - E_i) + \lambda \ D_i, \qquad (3)$$

where $\lambda$ defines the tradeoff between accuracy and diversity. Finally, ADDEMUP prunes the population to the $N$ most-fit members, which it defines to be its current ensemble, then repeats this process.

We define our accuracy term, $1 - E_i$, to be network $i$'s validation-set accuracy (or training-set accuracy if a validation set is not used), and we use Equation 1 over this validation set to calculate our diversity term $D_i$. We then separately normalize each term so that the values range from 0 to 1. Normalizing both terms allows $\lambda$ to have the same meaning across domains. Since it is not always clear at what value one should set $\lambda$, we have therefore developed some rules for automatically setting $\lambda$. First, we never change $\lambda$ if the ensemble error $\hat{E}$ is decreasing while we consider new networks; otherwise we change $\lambda$ if one of following two things happen: (1) population error $\bar{E}$ is not increasing and the population diversity $\bar{D}$ is decreasing; diversity seems to be under-emphasized and we increase $\lambda$, or (2) $\bar{E}$ is increasing and $\bar{D}$ is not decreasing; diversity seems to be over-emphasized and we decrease $\lambda$. (We started $\lambda$ at 0.1 for the results in this paper.)

A useful network to add to an ensemble is one that correctly classifies as many examples as possible while making its mistakes primarily on examples that most

Table 1: The ADDEMUP algorithm.

**GOAL:** Genetically create an accurate and diverse ensemble of networks.

1. Create and train the initial population of networks.
2. Until a stopping criterion is reached:
   (a) Use genetic operators to create new networks.
   (b) Train the new networks using Equation 4 and add them to the population.
   (c) Measure the diversity of each network with respect to the current population (see Equation 1).
   (d) Normalize the accuracy scores and the diversity scores of the individual networks.
   (e) Calculate the fitness of each population member (see Equation 3).
   (f) Prune the population to the $N$ fittest networks.
   (g) Adjust $\lambda$ (see the text for an explanation).
   (h) Report the current population of networks as the ensemble. Combine the output of the networks according to Equation 5.

---

of the current population members correctly classify. We address this during backpropagation training by multiplying the usual cost function by a term that measures the combined population error on that example:

$$
Cost = \sum_{k \in T} \left| \frac{t(k) - \hat{o}(k)}{\hat{E}} \right|^{\frac{\lambda}{\lambda+1}} [t(k) - a(k)]^2, \tag{4}
$$

where $t(k)$ is the target and $a(k)$ is the network activation for example $k$ in the training set $T$. Notice that since our network is not yet a member of the ensemble, $\hat{o}(k)$ and $\hat{E}$ are not dependent on our network; our new term is thus a constant when calculating the derivatives during backpropagation. We normalize $t(k) - \hat{o}(k)$ by the ensemble error $\hat{E}$ so that the *average* value of our new term is around 1 regardless of the correctness of the ensemble. This is especially important with highly accurate populations, since $t_k - \hat{o}(k)$ will be close to 0 for most examples, and the network would only get trained on a few examples. The exponent $\frac{\lambda}{\lambda+1}$ represents the ratio of importance of the diversity term in the fitness function. For instance, if $\lambda$ is close to 0, diversity is not considered important and the network is trained with the usual cost function; however, if $\lambda$ is large, diversity is considered important and our new term in the cost function takes on more importance.

We combine the predictions of the networks by taking a weighted sum of the output of each network, where each weight is based on the validation-set accuracy of the network. Thus we define our weights for combining the networks as follows:

$$
w_i = \frac{1 - E_i}{\sum_k (1 - E_k)} \tag{5}
$$

While simply averaging the outputs generates a good composite model (Clemen, 1989), we include the predicted accuracy in our weights since one should believe accurate models more than inaccurate ones.

## 4  Experimental Study

The genetic algorithm we use for generating new network topologies is the RE-GENT algorithm (Opitz and Shavlik, 1994). REGENT uses genetic algorithms to search through the space of *knowledge-based neural network* (KNN) topologies. KNNs are networks whose topologies are determined as a result of the direct mapping of a set of background rules that represent what we currently know about our task. KBANN (Towell and Shavlik, 1994), for instance, translates a set of propositional rules into a neural network, then refines the resulting network's weights using backpropagation. Trained KNNs, such as KBANN's networks, have been shown to frequently generalize better than many other inductive-learning techniques such as standard neural networks (Opitz, 1995; Towell and Shavlik, 1994). Using KNNs allows us to have highly correct networks in our ensemble; however, since each network in our ensemble is initialized with the same set of domain-specific rules, we do not expect there to be much disagreement among the networks. An alternative we consider in our experiments is to randomly generate our initial population of network topologies, since domain-specific rules are sometimes not available.

We ran ADDEMUP on NYNEX's MAX problem set and on three problems from the Human Genome Project that aid in locating genes in DNA sequences (recognizing *promoters*, *splice-junctions*, and *ribosome-binding sites - RBS*). Each of these domains is accompanied by a set of approximately correct rules describing what is currently known about the task (see Opitz, 1995 or Opitz and Shavlik, 1994 for more details). Our experiments measure the test-set error of ADDEMUP on these tasks. Each ensemble consists of 20 networks, and the REGENT and ADDEMUP algorithms considered 250 networks during their genetic search.

Table 2a presents the results from the case where the learners randomly create the topology of their networks (i.e., they do not use the domain-specific knowledge). Table 2a's first row, best-network, results from a single-layer neural network where, for each fold, we trained 20 networks containing between 0 and 100 (uniformly) hidden nodes and used a validation set to choose the best network. The next row, bagging, contains the results of running Breiman's (1994) *bagging* algorithm on standard, single-hidden-layer networks, where the number of hidden nodes is randomly set between 0 and 100 for each network.[2] Bagging is a "bootstrap" ensemble method that trains each network in the ensemble with a different partition of the training set. It generates each partition by randomly drawing, with replacement, $N$ examples from the training set, where $N$ is the size of the training set. Breiman (1994) showed that bagging is effective on "unstable" learning algorithms, such as neural networks, where small changes in the training set result in large changes in predictions. The bottom row of Table 2a, ADDEMUP, contains the results of a run of ADDEMUP where its initial population (of size 20) is randomly generated. The results show that on these domains combining the output of multiple trained networks generalizes better than trying to pick the single-best network.

While the top table shows the power of neural-network ensembles, Table 2b demonstrates ADDEMUP's ability to utilize prior knowledge. The first row of Table 2b contains the generalization results of the KBANN algorithm, while the next row, KBANN-bagging, contains the results of the ensemble where each individual network in the ensemble is the KBANN network trained on a different partition of the training set. Even though each of these networks start with the same topology and

Table 2: Test-set error from a ten-fold cross validation. Table (a) shows the results from running three learners without the domain-specific knowledge; Table (b) shows the results of running three learners with this knowledge. Pairwise, one-tailed *t*-tests indicate that ADDEMUP in Table (b) differs from the other algorithms in both tables at the 95% confidence level, except with REGENT in the splice-junction domain.

| Standard neural networks (no domain-specific knowledge used) | | | | |
|---|---|---|---|---|
| | Promoters | Splice Junction | RBS | MAX |
| best-network | 6.6% | 7.8% | 10.7% | 37.0% |
| bagging | 4.6% | 4.5% | 9.5% | 35.7% |
| ADDEMUP | 4.6% | 4.9% | 9.0% | 34.9% |

(a)

| Knowledge-based neural networks (domain-specific knowledge used) | | | | |
|---|---|---|---|---|
| | Promoters | Splice Junction | RBS | MAX |
| KBANN | 6.2% | 5.3% | 9.4% | 35.8% |
| KBANN-bagging | 4.2% | 4.5% | 8.5% | 35.6% |
| REGENT-combined | 3.9% | 3.9% | 8.2% | 35.6% |
| ADDEMUP | 2.9% | 3.6% | 7.5% | 34.7% |

(b)

"large" initial weight settings (i.e., the weights resulting from the domain-specific knowledge), small changes in the training set still produce significant changes in predictions. Also notice that on all datasets, KBANN-bagging is as good as or better than running bagging on randomly generated networks (i.e., bagging in Table 2a).

The next row, REGENT-combined, contains the results of simply combining, using Equation 5, the networks in REGENT's final population. ADDEMUP, the final row of Table 2b, mainly differs from REGENT-combined in two ways: (a) its fitness function (i.e., Equation 3) takes into account diversity rather than just network accuracy, and (b) it trains new networks by emphasizing the erroneous examples of the current ensemble. Therefore, comparing ADDEMUP with REGENT-combined helps directly test ADDEMUP's diversity-achieving heuristics, though additional results reported in Opitz (1995) show ADDEMUP gets *most* of its improvement from its fitness function. There are two main reasons why we think the results of ADDEMUP in Table 2b are especially encouraging: (a) by comparing ADDEMUP with REGENT-combined, we explicitly test the quality of our heuristics and demonstrate their effectiveness, and (b) ADDEMUP is able to effectively utilize background knowledge to decrease the error of the individual networks in its ensemble, while still being able to create enough diversity among them so as to improve the overall quality of the ensemble.

## 5  Conclusions

Previous work with neural-network ensembles have shown them to be an effective technique if the classifiers in the ensemble are both highly correct and disagree with each other as much as possible. Our new algorithm, ADDEMUP, uses genetic algorithms to search for a correct and diverse population of neural networks to be used in the ensemble. It does this by collecting the set of networks that best fits an objective function that measures both the accuracy of the network and the disagreement of that network with respect to the other members of the set. ADDEMUP tries

to actively generate quality networks during its search by emphasizing the current ensemble's erroneous examples during backpropagation training.

Experiments demonstrate that our method is able to find an effective set of networks for our ensemble. Experiments also show that ADDEMUP is able to effectively incorporate prior knowledge, if available, to improve the quality of this ensemble. In fact, when using domain-specific rules, our algorithm showed statistically significant improvements over (a) the single best network seen during the search, (b) a previously proposed ensemble method called bagging (Breiman, 1994), and (c) a similar algorithm whose objective function is simply the validation-set correctness of the network. In summary, ADDEMUP is successful in generating a set of neural networks that work well together in producing an accurate prediction.

## Acknowledgements

This work was supported by Office of Naval Research grant N00014-93-1-0998.

## Footnotes

[1]Krogh and Vedelsby referred to this term as *ambiguity*.

[2]We also tried other ensemble approaches, such as randomly creating varying multilayer network topologies and initial weight settings, but bagging did significantly better on all datasets (by 15-25% on all three DNA domains).

# References

Alpaydin, E. (1993). Multiple networks for function learning. In *Proceedings of the 1993 IEEE International Conference on Neural Networks*, vol I, pages 27–32, San Fransisco.

Breiman, L. (1994). Bagging predictors. Technical Report 421, Department of Statistics, University of California, Berkeley.

Clemen, R. (1989). Combining forecasts: A review and annotated bibliography. *International Journal of Forecasting*, 5:559–583.

Drucker, H., Cortes, C., Jackel, L., LeCun, Y., and Vapnik, V. (1994). Boosting and other machine learning algorithms. In *Proceedings of the Eleventh International Conference on Machine Learning*, pages 53–61, New Brunswick, NJ. Morgan Kaufmann.

Hansen, L. and Salamon, P. (1990). Neural network ensembles. *IEEE Transactions on Pattern Analysis and Machine Intelligence*, 12:993–1001.

Hashem, S., Schmeiser, B., and Yih, Y. (1994). Optimal linear combinations of neural networks: An overview. In *Proceedings of the 1994 IEEE International Conference on Neural Networks*, Orlando, FL.

Krogh, A. and Vedelsby, J. (1995). Neural network ensembles, cross validation, and active learning. In Tesauro, G., Touretzky, D., and Leen, T., editors, *Advances in Neural Information Processing Systems*, vol 7, Cambridge, MA. MIT Press.

Maclin, R. and Shavlik, J. (1995). Combining the predictions of multiple classifiers: Using competitive learning to initialize neural networks. In *Proceedings of the Fourteenth International Joint Conference on Artificial Intelligence*, Montreal, Canada.

Mani, G. (1991). Lowering variance of decisions by using artificial neural network portfolios. *Neural Computation*, 3:484–486.

Opitz, D. (1995). *An Anytime Approach to Connectionist Theory Refinement: Refining the Topologies of Knowledge-Based Neural Networks*. PhD thesis, Computer Sciences Department, University of Wisconsin, Madison, WI.

Opitz, D. and Shavlik, J. (1994). Using genetic search to refine knowledge-based neural networks. In *Proceedings of the Eleventh International Conference on Machine Learning*, pages 208–216, New Brunswick, NJ. Morgan Kaufmann.

Perrone, M. (1992). A soft-competitive splitting rule for adaptive tree-structured neural networks. In *Proceedings of the International Joint Conference on Neural Networks*, pages 689–693, Baltimore, MD.

Towell, G. and Shavlik, J. (1994). Knowledge-based artificial neural networks. *Artificial Intelligence*, 70(1,2):119–165.

Wolpert, D. (1992). Stacked generalization. *Neural Networks*, 5:241–259.